# Using Genetic Algorithms to Improve Pattern Classification Performance

**Eric I. Chang** and **Richard P. Lippmann**
Lincoln Laboratory, MIT
Lexington, MA 02173-9108

## Abstract

Genetic algorithms were used to select and create features and to select reference exemplar patterns for machine vision and speech pattern classification tasks. For a complex speech recognition task, genetic algorithms required no more computation time than traditional approaches to feature selection but reduced the number of input features required by a factor of five (from 153 to 33 features). On a difficult artificial machine-vision task, genetic algorithms were able to create new features (polynomial functions of the original features) which reduced classification error rates from 19% to almost 0%. Neural net and k nearest neighbor (KNN) classifiers were unable to provide such low error rates using only the original features. Genetic algorithms were also used to reduce the number of reference exemplar patterns for a KNN classifier. On a 338 training pattern vowel-recognition problem with 10 classes, genetic algorithms reduced the number of stored exemplars from 338 to 43 without significantly increasing classification error rate. In all applications, genetic algorithms were easy to apply and found good solutions in many fewer trials than would be required by exhaustive search. Run times were long, but not unreasonable. These results suggest that genetic algorithms are becoming practical for pattern classification problems as faster serial and parallel computers are developed.

## 1 INTRODUCTION

Feature selection and creation are two of the most important and difficult tasks in the field of pattern classification. Good features improve the performance of both conventional and neural network pattern classifiers. Exemplar selection is another task that can reduce the memory and computation requirements of a KNN classifier. These three tasks require a search through a space which is typically so large that

exhaustive search is impractical. The purpose of this research was to explore the usefulness of Genetic search algorithms for these tasks. Details concerning this research are available in (Chang, 1990).

Genetic algorithms depend on the generation-by-generation development of possible solutions, with selection eliminating bad solutions and allowing good solutions to replicate and be modified. There are four stages in the genetic search process: creation, selection, crossover, and mutation. In the creation stage, a group of possible solutions to a search problem is randomly generated. In most genetic algorithm applications, each solution is a bit string with each bit initially randomly set to 1 or 0.

After the creation stage, each solution is evaluated using a fitness function and assigned a fitness value. The fitness function must be tightly linked to the eventual goal. The usual criterion for success in pattern classification tasks is the percentage of patterns classified correctly on test data. This was approximated in all experiments by using a leave-one-out cross-validation measure of classification accuracy obtained using training data and a KNN classifier. After solutions are assigned fitness values, a selection stage occurs, where the fitter solutions are given more chance to reproduce. This gives the fitter solutions more and more influence over the changes in the population so that eventually fitter solutions dominate.

A crossover operation occurs after two fitter solutions (called parent solutions) have been selected. During crossover, portions of the parent solutions are exchanged. This operation is performed in the hope of generating new solutions which will contain the useful parts of both parent solutions and be even better solutions. Crossover is responsible for generating most of the new solutions in genetic search. When all solutions are similar, the crossover operation loses its ability to generate new solutions since exchanging portions of identical solutions generates the same solutions. Mutation (randomly altering bits) is performed on each new solution to prevent the whole population from becoming similar. However, mutation does not generally improve solutions by itself. The combination of both crossover and mutation is required for good performance.

There are many varieties of genetic algorithms. A relatively new incremental static population model proposed by (Whitley, 1989) was used in all experiments. In the regular genetic algorithm model, the whole population undergoes selection and reproduction, with a large portion of the strings replaced by new strings. It is thus possible for good strings to be deleted from the population. In the static population model, the population is ranked according to fitness. At each recombination cycle, two strings are picked as parents according to their fitness values, and two new strings are produced. These two new strings replace the lowest ranked strings in the original population. This model automatically protects the better strings in the population.

## 2   FEATURE SELECTION

Adding more input features or input dimensions to a pattern classifier often degrades rather than improves performance. This is because as the number of input features increases, the number of training patterns required to maintain good generalization and adequately describe class distributions also often increases rapidly. Performance with limited training data may thus degrade. Feature selection (dimensionality

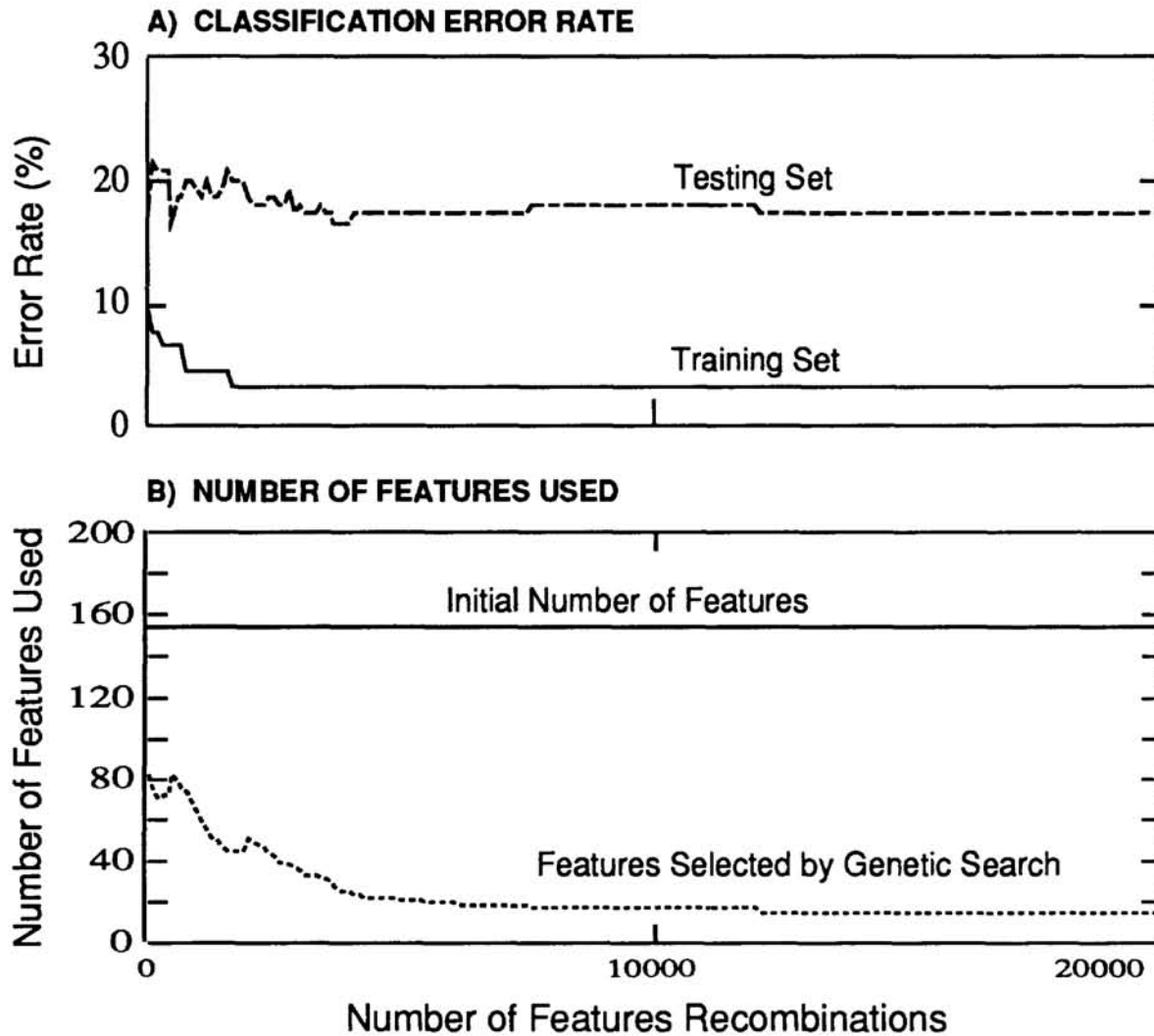

Figure 1: Progress Of a Genetic Algorithm Search For Those Features From an Original 153 Features That Provide High Accuracy in "E" Set Classification For One Female Talker: (A) Classification Error Rate and (B) Number Of Features Used.

reduction) is often required when training data is limited to select the subset of features that best separates classes. It can improve performance and/or reduce computation requirements.

Feature selection is difficult because the number of possible combinations of features grows exponentially with the number of original features. For a moderate size problem with 64 features, there are $2^{64}$ possible subsets of features. Clearly an exhaustive evaluation of each possible combination is impossible. Frequently, finding a near optimal feature subset is adequate. An overview of many different approaches to feature selection is available in (Siedlecki and Sklansky, 1988).

This work applies genetic search techniques to the problem of feature selection. Every feature set is represented by a bit string with $d$ bits, where $d$ is the maximum

input dimension. Each bit determines whether a feature is used. The accuracy of a KNN classifier with the leave-one-out approach to error rate estimation was used as an evaluation function as described above. A KNN classifier has the advantage of requiring no training time and providing results directly related to performance.

"E-set" words (9 letters from the English alphabet that rhyme with the letter "E") taken from a Texas Instruments 46-word speech database were used for experiments. Waveforms were spectrally analyzed and encoded with a hidden Markov Model speech recognizer as described in (Huang and Lippmann, 1990). Features were the average log likelihood distance and duration from all the hidden Markov nodes determined using Viterbi decoding. The final output of the hidden Markov model was also included in the feature set. This resulted in 17 features per word class. The 9 different word classes result in a total of 153 features. For each talker there were 10 patterns in the training set and 16 patterns in the testing set per word class. All experiments were talker dependent.

An experiment was performed using the data from one female talker. More conventional sequential forward and backward searches for the best feature subset were first performed. The total number of KNN evaluations for each sequential search was 11,781. The best feature subset found with sequential searches contained 33 features and the classification error rates were 2.2% and 18.5% on training and testing sets respectively. Genetic algorithms provided a lower error rate on the testing set with fewer than half as many features. Fig. 1 shows the progress of the genetic search. The bottom plot shows that near recombination 12,100, the number of features used was reduced to 15. The top plot shows that classification error rates were 3.3% and 17.5% for the training and testing sets respectively.

## 3   FEATURE CREATION

One of the most successful techniques for improving pattern classification performance with limited training data is to find more effective input features. An approach to creating more effective input features is to search through new features that are polynomial functions of the original features. This difficult search problem was explored using genetic algorithms. The fitness function was again determined using the performance of a KNN classifier with leave-one-out testing.

Polynomial functions of the original features taken two at a time were created as new features. New features were represented by a bit string consisting of substrings identifying the original features used, their exponents, and the operation to be applied between the original features. A gradual buildup of feature complexity over multiple stages was enforced by limiting the complexity of the created features. Once the accuracy of a KNN classifier had converged at one stage, another stage was begun where more complex high order features were allowed. This improves generalization by creating simple features first and by creating more complicated features only when simpler features are not satisfactory.

A parallel vector problem, where the input data consists of $\Delta x, \Delta y$ of two vectors, was used. Parallel vectors are identified as one class while nonparallel vectors are identified as another. There were 300 training patterns and 100 testing patterns. During an experiment, the ratio features $\Delta x2/\Delta x1$ and $\Delta y2/\Delta y1$ were first found near recombination 700. After the error rate had not changed for 2,000 recombinations, the complexity of the created features was allowed to increase at recombina-

tion 2,700. At this point the two ratio features and the four original features were treated as if they were six original features. The final feature found after this point was $(\triangle x_2 * \triangle y_2)/(\triangle x_1 * \triangle y_1)$. Classification error rates for the training set and the testing set decreased to 0% with this feature. The classification error rate on the testing set using the original four features was 19% using a KNN classifier. Tests using the original features with two more complex classifiers also used in (Ng and Lippmann, 1991) resulted in error rates of 13.3% for a GMDH classifier and 8.3% for a radial basis function classifier. Feature creation with a simple KNN classifier was thus more effective than the use of more complex classifiers with the original features.

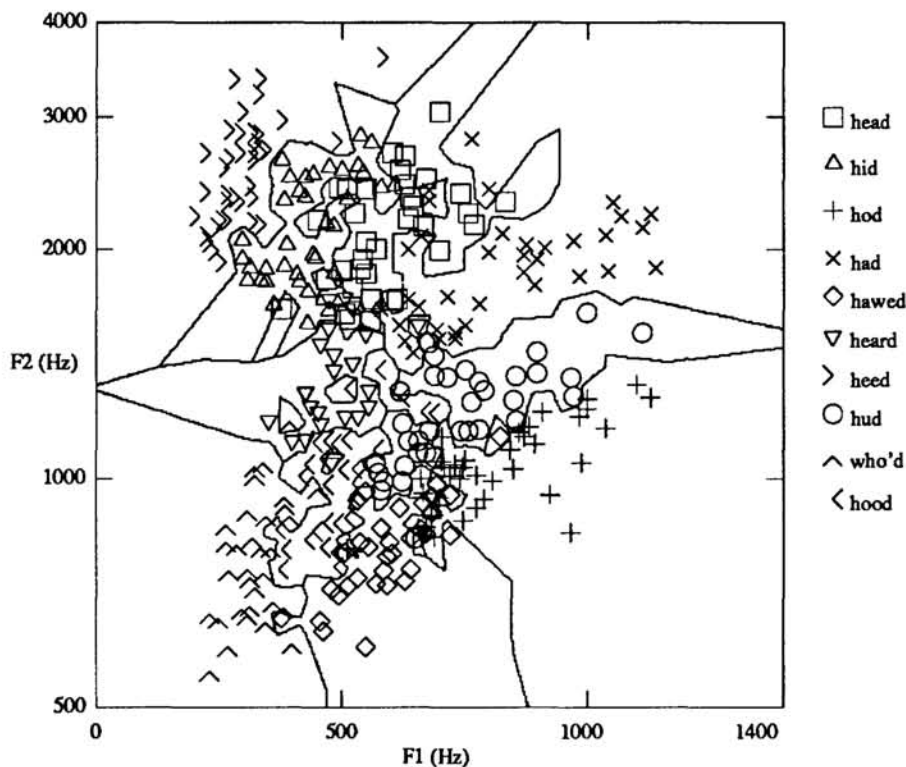

Figure 2: Decision Boundaries Of a Nearest Neighbor Classifier For The Vowel Problem Using All 338 Original Exemplars.

## 4    EXEMPLAR SELECTION

The performance of a KNN classifier typically improves as more training patterns are stored. This often makes KNN classifiers impractical because both classification time and memory requirements increase linearly with the number of training patterns. Previous approaches to reducing the classification time and memory requirements of KNN classifiers include using KD trees and condensed k nearest neighbor (CKNN) classifiers as described in (Ng and Lippmann, 1991). KD trees, however, are effective only if the input dimensionality is low, and CKNN classifiers use a heuristic that may not result in minimal memory requirements. An alternate

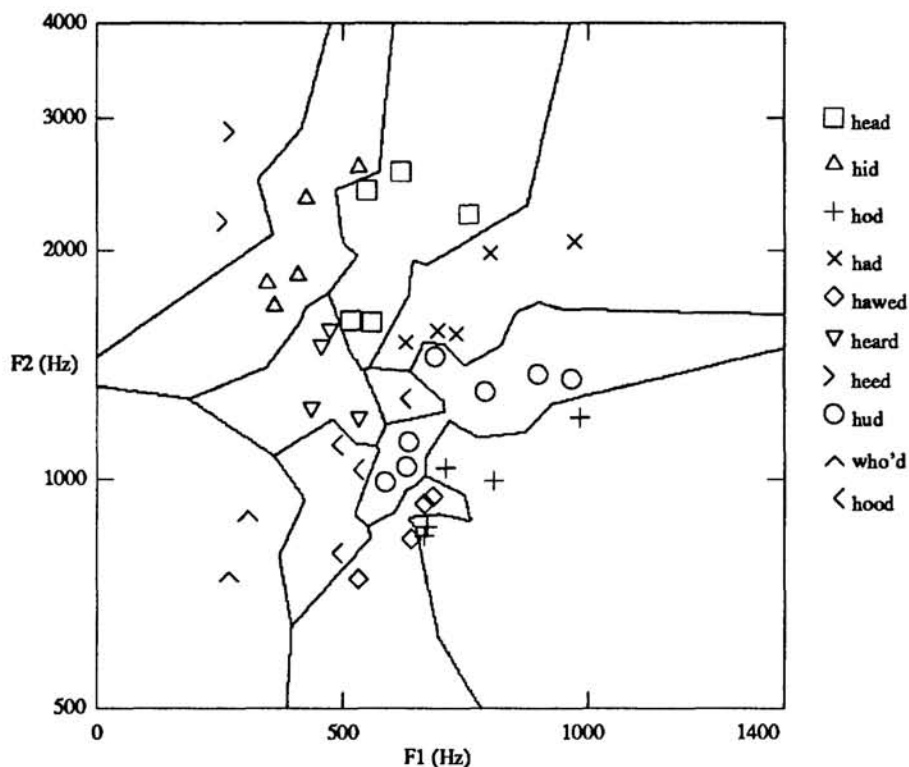

Figure 3: Decision Boundaries Of a Nearest Neighbor Classifier For The Vowel Problem Using 43 Exemplars selected Using Genetic Search.

approach is to use genetic algorithms.

Genetic algorithms were used to manipulate bit strings identifying useful exemplar patterns. A bonus proportional to the number of unused exemplars was given to strings with classifier accuracy above a user-preset threshold. The value $k$ was also selected by genetic algorithms in some experiments. The $k$ value was encoded with three bits which were attached to the end of each string. Exemplar selection was tested with the vowel database used by (Ng and Lippmann, 1991). There were ten classes, each class being a word starting with "h" and ending with "d", with a vowel in between ("head", "hid", "hod", "had", "hawed", "heard", "heed", "hud", "who'd", and "hood"). A total of 338 patterns was used as the training set and 333 patterns were used as the testing set. Each pattern consisted of two features which were the two formant frequencies of the vowel determined by spectrographic analysis.

Genetic algorithms were effective in both reducing the number of exemplars and selecting $k$. Classification error rates with selected exemplars were roughly 20% on both training and test data. Selecting $k$ typically resulted in fewer exemplars and the number of exemplars required was reduced by a factor of roughly 8 (from 338 to 43). Genetic search was thus much more effective than the CKNN classifier

described in (Ng and Lippmann, 1991) which reduced the number of exemplars by a factor of roughly 2 (from 338 to 152). Decision boundaries with all 338 original exemplars are shown in Fig. 2. Boundaries are excessively complex and provide perfect performance on the training patterns but perform poorly on the testing patterns (25% error rate). Decision boundaries with the 43 exemplars selected using genetic algorithms are shown in Fig. 3. Boundaries with the smaller number of exemplars are smoother and provide an error rate of 20.1% on test data.

## 5    CONCLUSIONS

Genetic algorithms proved to be a good search technique which is widely applicable in pattern classification. Genetic algorithms were relatively easy to apply to feature selection, feature creation, and exemplar selection problems. Solutions were found that were better than those provided by heuristic approaches including forward and backward feature selection and condensed k nearest neighbor algorithms. Genetic algorithms also required far fewer evaluations than required by exhaustive search and sometimes required only little more computation than heuristic approaches. Run times on a Sun-3 workstation were long (hours and sometimes one or two days) but not impractical. Run times are becoming become less of an issue as single-processor workstations become more powerful and as parallel computers become more available. Compared to developing a heuristic search technique for each type of search problem, genetic algorithms offer the benefit of simplicity and good performance on all problems. Further experiments should explore the use of genetic algorithms in other application areas and also compare alternative search techniques including simulated annealing.

## Acknowledgements

This work was sponsored by the Air Force Office of Scientific Research and the Department of the Air Force.

## References

Eric I. Chang. Using Genetic Algorithms to Select and Create Features for Pattern Classification. Master's Thesis, Massachusetts Institute of Technology, Department of Electrical Engineering and Computer Science, Cambridge, MA, May 1990.

William Y. Huang and Richard P. Lippmann. HMM Speech Recognition Systems with Neural Net Discrimination. In D. Touretzky (Ed.) *Advances in Neural Information Processing Systems 2*, 194–202, 1990.

Kenney Ng and Richard P. Lippmann. A Comparative Study of the Practical Characteristics of Neural Network and Conventional Pattern Classifiers. In Lippmann, R., Moody, J., Touretzky, D., (Eds.) *Advances in Neural Information Processing Systems 3*, 1991.

W. Siedlecki and J. Sklansky. On Automatic Feature Selection. *International Journal of Pattern Recognition and Artificial Intelligence*, 2:197–220, 1988.

Darrel Whitley. The GENITOR Algorithm and Selection Pressure: Why Rank-Based Allocation of Reproductive Trials is Best. In *Proceedings Third International Conference on Genetic Algorithms*, Washington, DC, June 1989.